# Real Time Voice Processing with Audiovisual Feedback: Toward Autonomous Agents with Perfect Pitch

**Lawrence K. Saul**[1]**, Daniel D. Lee**[2]**, Charles L. Isbell**[3]**, and Yann LeCun**[4]

[1] Department of Computer and Information Science
[2]Department of Electrical and System Engineering
University of Pennsylvania, 200 South 33rd St, Philadelphia, PA 19104
[3]Georgia Tech College of Computing, 801 Atlantic Drive, Atlanta, GA 30332
[4]NEC Research Institute, 4 Independence Way, Princeton, NJ 08540
lsaul@cis.upenn.edu, ddlee@ee.upenn.edu, isbell@cc.gatech.edu, yann@research.nj.nec.com

## Abstract

We have implemented a real time front end for detecting voiced speech and estimating its fundamental frequency. The front end performs the signal processing for voice-driven agents that attend to the pitch contours of human speech and provide continuous audiovisual feedback. The algorithm we use for pitch tracking has several distinguishing features: it makes no use of FFTs or autocorrelation at the pitch period; it updates the pitch incrementally on a sample-by-sample basis; it avoids peak picking and does not require interpolation in time or frequency to obtain high resolution estimates; and it works reliably over a four octave range, in real time, without the need for postprocessing to produce smooth contours. The algorithm is based on two simple ideas in neural computation: the introduction of a purposeful nonlinearity, and the error signal of a least squares fit. The pitch tracker is used in two real time multimedia applications: a voice-to-MIDI player that synthesizes electronic music from vocalized melodies, and an audiovisual Karaoke machine with multimodal feedback. Both applications run on a laptop and display the user's pitch scrolling across the screen as he or she sings into the computer.

## 1 Introduction

The pitch of the human voice is one of its most easily and rapidly controlled acoustic attributes. It plays a central role in both the production and perception of speech[17]. In clean speech, and even in corrupted speech, pitch is generally perceived with great accuracy[2, 6] at the fundamental frequency characterizing the vibration of the speaker's vocal chords.

There is a large literature on machine algorithms for pitch tracking[7], as well as applications to speech synthesis, coding, and recognition. Most algorithms have one or more of the following components. First, sliding windows of speech are analyzed at 5-10 ms intervals, and the results concatenated over time to obtain an initial estimate of the pitch contour. Second, within each window (30-60 ms), the pitch is deduced from peaks in the

windowed autocorrelation function[13] or power spectrum[9, 10, 15], then refined by further interpolation in time or frequency. Third, the estimated pitch contours are smoothed by a postprocessing procedure[16], such as dynamic programming or median filtering, to remove octave errors and isolated glitches.

In this paper, we describe an algorithm for pitch tracking that works quite differently and—based on our experience—quite well as a real time front end for interactive voice-driven agents. Notably, our algorithm does not make use of FFTs or autocorrelation at the pitch period; it updates the pitch incrementally on a sample-by-sample basis; it avoids peak picking and does not require interpolation in time or frequency to obtain high resolution estimates; and it works reliably over a four octave range—in real time—without any post-processing. We have implemented the algorithm in two real-time multimedia applications: a voice-to-MIDI player and an audiovisual Karaoke machine. More generally, we are using the algorithm to explore novel types of human-computer interaction, as well as studying extensions of the algorithm for handling corrupted speech and overlapping speakers.

## 2  Algorithm

A pitch tracker performs two essential functions: it labels speech as voiced or unvoiced, and throughout segments of voiced speech, it computes a running estimate of the fundamental frequency. Pitch tracking thus depends on the running detection and identification of periodic signals in speech. We develop our algorithm for pitch tracking by first examining the simpler problem of detecting sinusoids. For this simpler problem, we describe a solution that does not involve FFTs or autocorrelation at the period of the sinusoid. We then extend this solution to the more general problem of detecting periodic signals in speech.

### 2.1  Detecting sinusoids

A simple approach to detecting sinusoids is based on viewing them as the solution of a second order linear difference equation[12]. A discretely sampled sinusoid has the form:

$$s_n = A \sin(\omega n + \theta). \tag{1}$$

Sinusoids obey a simple difference equation such that each sample $s_n$ is proportional to the average of its neighbors $\frac{1}{2}(s_{n-1} + s_{n+1})$, with the constant of proportionality given by:

$$s_n = (\cos \omega)^{-1} \left[ \frac{s_{n-1} + s_{n+1}}{2} \right]. \tag{2}$$

Eq. (2) can be proved using trigonometric identities to expand the terms on the right hand side. We can use this property to judge whether an unknown signal $x_n$ is approximately sinusoidal. Consider the error function:

$$\mathcal{E}(\alpha) = \sum_n \left[ x_n - \alpha \left( \frac{x_{n-1} + x_{n+1}}{2} \right) \right]^2. \tag{3}$$

If the signal $x_n$ is well described by a sinusoid, then the right hand side of this error function will achieve a small value when the coefficient $\alpha$ is tuned to match its frequency, as in eq. (2). The minimum of the error function is found by solving a least squares problem:

$$\alpha^* = \frac{2 \sum_n x_n (x_{n-1} + x_{n+1})}{\sum_n (x_{n-1} + x_{n+1})^2}. \tag{4}$$

Thus, to test whether a signal $x_n$ is sinusoidal, we can minimize its error function by eq. (4), then check two conditions: first, that $\mathcal{E}(\alpha^*) \ll \mathcal{E}(0)$, and second, that $|\alpha^*| \geq 1$. The first condition establishes that the mean squared error is small relative to the mean

squared amplitude of the signal, while the second establishes that the signal is sinusoidal (as opposed to exponential), with estimated frequency:

$$\omega^* \;=\; \cos^{-1}(1/\alpha^*). \tag{5}$$

This procedure for detecting sinusoids (known as Prony's method[12]) has several notable features. First, it does not rely on computing FFTs or autocorrelation at the period of the sinusoid, but only on computing the zero-lagged and one-sample-lagged autocorrelations that appear in eq. (4), namely $\sum_n x_n^2$ and $\sum_n x_n x_{n\pm1}$. Second, the frequency estimates are obtained from the solution of a least squares problem, as opposed to the peaks of an autocorrelation or FFT, where the resolution may be limited by the sampling rate or signal length. Third, the method can be used in an incremental way to track the frequency of a slowly modulated sinusoid. In particular, suppose we analyze sliding windows—shifted by *just one sample at a time*—of a longer, nonstationary signal. Then we can efficiently update the windowed autocorrelations that appear in eq. (4) by adding just those terms generated by the rightmost sample of the current window and dropping just those terms generated by the leftmost sample of the previous window. (The number of operations per update is constant and does not depend on the window size.)

We can extract more information from the least squares fit besides the error in eq. (3) and the estimate in eq. (5). In particular, we can characterize the *uncertainty* in the estimated frequency. The normalized error function $\mathcal{N}(\alpha)\!=\!\log[\mathcal{E}(\alpha)/\mathcal{E}(0)]$ evaluates the least squares fit on a dimensionless logarithmic scale that does not depend on the amplitude of the signal. Let $\mu\!=\!\log(\cos^{-1}(1/\alpha))$ denote the *log-frequency* implied by the coefficient $\alpha$, and let $\Delta\mu^*$ denote the uncertainty in the estimated log-frequency $\mu^* = \log\omega^*$. (By working in the log domain, we measure uncertainty in the same units as the distance between notes on the musical scale.) A heuristic measure of uncertainty is obtained by evaluating the *sharpness* of the least squares fit, as characterized by the second derivative:

$$\Delta\mu^* \;=\; \left[\left(\frac{\partial^2 \mathcal{N}}{\partial\mu^2}\right)\bigg|_{\mu=\mu^*}\right]^{-\frac{1}{2}} = \frac{1}{\omega^*}\left(\frac{\cos^2\omega^*}{\sin\omega^*}\right)\left[\left(\frac{1}{\mathcal{E}}\frac{\partial^2\mathcal{E}}{\partial\alpha^2}\right)\bigg|_{\alpha=\alpha^*}\right]^{-\frac{1}{2}}. \tag{6}$$

Eq. (6) relates sharper fits to lower uncertainty, or higher precision. As we shall see, it provides a valuable criterion for comparing the results of different least squares fits.

## 2.2 Detecting voiced speech

Our algorithm for detecting voice speech is a simple extension of the algorithm described in the previous section. The algorithm operates on the time domain waveform in a number of stages, as summarized in Fig. 1. The analysis is based on the assumption that the low frequency spectrum of voiced speech can be modeled as a sum of (noisy) sinusoids occurring at integer multiples of the fundamental frequency, $f_0$.

*Stage 1. Lowpass filtering*
The first stage of the algorithm is to lowpass filter the speech, removing energy at frequencies above 1 kHz. This is done to eliminate the aperiodic component of voiced fricatives[17], such as /z/. The signal can be aggressively downsampled after lowpass filtering, though the sampling rate should remain at least twice the maximum allowed value of $f_0$. The lower sampling rate determines the rate at which the estimates of $f_0$ are updated, but it does not limit the resolution of the estimates themselves. (In our formal evaluations of the algorithm, we downsampled from 20 kHz to 4 kHz after lowpass filtering; in the real-time multimedia applications, we downsampled from 44.1 kHz to 3675 Hz.)

*Stage 2. Pointwise nonlinearity*
The second stage of the algorithm is to pass the signal through a pointwise nonlinearity, such as squaring or half-wave rectification (which clips negative samples to zero). The

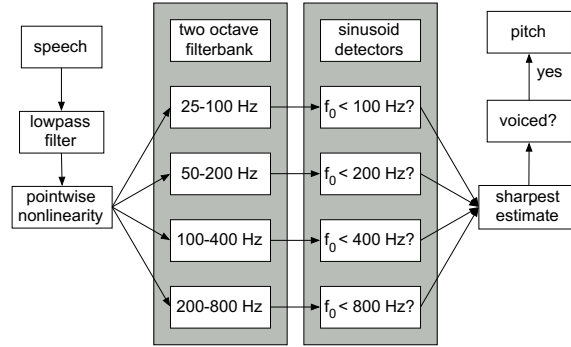

Figure 1: Estimating the fundamental frequency $f_0$ of voiced speech without FFTs or auto-correlation at the pitch period. The speech is lowpass filtered (and optionally downsampled) to remove fricative noise, then transformed by a pointwise nonlinearity that concentrates additional energy at $f_0$. The resulting signal is analyzed by a bank of bandpass filters that are narrow enough to resolve the harmonic at $f_0$, but too wide to resolve higher-order harmonics. A resolved harmonic at $f_0$ (essentially, a sinusoid) is detected by a running least squares fit, and its frequency recovered as the pitch. If more that one sinusoid is detected at the outputs of the filterbank, the one with the sharpest fit is used to estimate the pitch; if no sinusoid is detected, the speech is labeled as unvoiced. (The two octave filterbank in the figure is an idealization. In practice, a larger bank of narrower filters is used.)

purpose of the nonlinearity is to concentrate additional energy at the fundamental, particularly if such energy was missing or only weakly present in the original signal. In voiced speech, pointwise nonlinearities such as squaring or half-wave rectification tend to create energy at $f_0$ by virtue of extracting a crude representation of the signal's envelope. This is particularly easy to see for the operation of squaring, which—applied to the sum of two sinusoids—creates energy at their sum and difference frequencies, the latter of which characterizes the envelope. In practice, we use half-wave rectification as the nonlinearity in this stage of the algorithm; though less easily characterized than squaring, it has the advantage of preserving the dynamic range of the original signal.

*Stage 3. Filterbank*
The third stage of the algorithm is to analyze the transformed speech by a bank of bandpass filters. These filters are designed to satisfy two competing criteria. On one hand, they are sufficiently narrow to resolve the harmonic at $f_0$; on the other hand, they are sufficiently wide to integrate higher-order harmonics. An idealized two octave filterbank that meets these criteria is shown in Fig. 1. The result of this analysis—for voiced speech—is that the output of the filterbank consists either of sinusoids at $f_0$ (and not any other frequency), or signals that do not resemble sinusoids at all. Consider, for example, a segment of voiced speech with fundamental frequency $f_0 = 180$ Hz. For such speech, only the second filter from 50-200 Hz will resolve the harmonic at 180 Hz. On the other hand, the first filter from 25-100 Hz will pass low frequency noise; the third filter from 100-400 Hz will pass the first and second harmonics at 180 Hz and 360 Hz, and the fourth filter from 200-800 Hz will pass the second through fourth harmonics at 360, 540, and 720 Hz. Thus, the output of the filterbank will consist of a sinusoid at $f_0$ and three other signals that are random or periodic, but *definitely not sinusoidal*. In practice, we do not use the idealized two octave filterbank shown in Fig. 1, but a larger bank of narrower filters that helps to avoid contaminating the harmonic at $f_0$ by energy at $2f_0$. The bandpass filters in our experiments were 8th order Chebyshev (type I) filters with 0.5 dB of ripple in 1.6 octave passbands, and signals were doubly filtered to obtain sharp frequency cutoffs.

*Stage 4. Sinusoid detection*

The fourth stage of the algorithm is to detect sinusoids at the outputs of the filterbank. Sinusoids are detected by the adaptive least squares fits described in section 2.1. Running estimates of sinusoid frequencies and their uncertainties are obtained from eqs. (5–6) and updated on a sample by sample basis for the output of each filter. If the uncertainty in any filter's estimate is less than a specified threshold, then the corresponding sample is labeled as voiced, and the fundamental frequency $f_0$ determined by whichever filter's estimate has the least uncertainty. (For sliding windows of length 40–60 ms, the thresholds typically fall in the range 0.08–0.12, with higher thresholds required for shorter windows.) Empirically, we have found the uncertainty in eq. (6) to be a better criterion than the error function itself for evaluating and comparing the least squares fits from different filters. A possible explanation for this is that the expression in eq. (6) was derived by a dimensional analysis, whereas the error functions of different filters are not even computed on the same signals.

Overall, the four stages of the algorithm are well suited to a real time implementation. The algorithm can also be used for batch processing of waveforms, in which case startup and ending transients can be minimized by zero-phase forward and reverse filtering.

# 3  Evaluation

The algorithm was evaluated on a small database of speech collected at the University of Edinburgh[1]. The Edinburgh database contains about 5 minutes of speech consisting of 50 sentences read by one male speaker and one female speaker. The database also contains reference $f_0$ contours derived from simultaneously recorded larynogograph signals. The sentences in the database are biased to contain difficult cases for $f_0$ estimation, such as voiced fricatives, nasals, liquids, and glides. The results of our algorithm on the first three utterances of each speaker are shown in Fig. 2.

A formal evaluation was made by accumulating errors over all utterances in the database, using the reference $f_0$ contours as ground truth[1]. Comparisons between estimated and reference $f_0$ values were made every 6.4 ms, as in previous benchmarks. Also, in these evaluations, the estimates of $f_0$ from eqs. (4–5) were confined to the range 50–250 Hz for the male speaker and the range 120–400 Hz for the female speaker; this was done for consistency with previous benchmarks, which enforced these limits. Note that our estimated $f_0$ contours were *not* postprocessed by a smoothing procedure, such as median filtering or dynamic programming.

Error rates were computed for the fraction of unvoiced (or silent) speech misclassified as voiced and for the fraction of voiced speech misclassified as unvoiced. Additionally, for the fraction of speech correctly identified as voiced, a gross error rate was computed measuring the percentage of comparisons for which the reference and estimated $f_0$ differed by more than 20%. Finally, for the fraction of speech correctly identified as voiced and in which the estimated $f_0$, was not in gross error, a root mean square (rms) deviation was computed between the reference and estimated $f_0$.

The original study on this database published results for a number of approaches to pitch tracking. Earlier results, as well as those derived from the algorithm in this paper, are shown in Table 1. The overall results show our algorithm—indicated as the adaptive least squares (ALS) approach to pitch tracking—to be extremely competitive in all respects. The only anomaly in these results is the slightly larger rms deviation produced by ALS estimation compared to other approaches. The discrepancy could be an artifact of the filtering operations in Fig. 1, resulting in a slight desynchronization of the reference and estimated $f_0$ contours. On the other hand, the discrepancy could indicate that for certain voiced sounds, a more robust estimation procedure[12] would yield better results than the simple least squares fits in section 2.1.

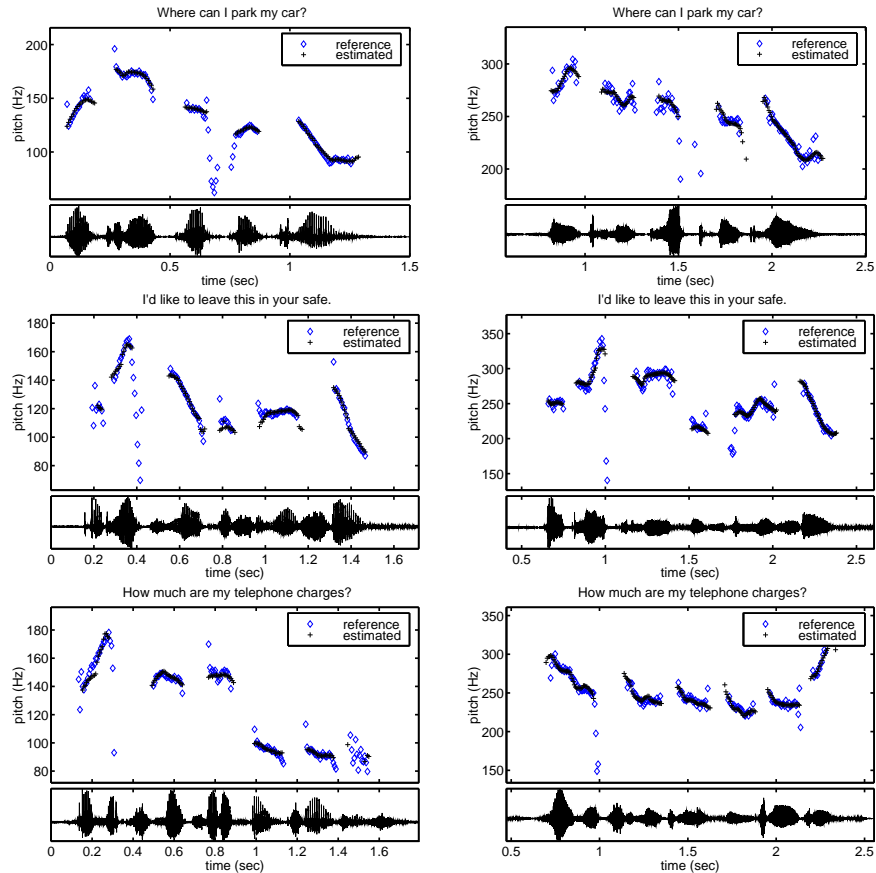

Figure 2: Reference and estimated $f_0$ contours for the first three utterances of the male (left) and female (right) speaker in the Edinburgh database[1]. Mismatches between the contours reveal voiced and unvoiced errors.

## 4 Agents

We have implemented our pitch tracking algorithm as a real time front end for two interactive voice-driven agents. The first is a voice-to-MIDI player that synthesizes electronic music from vocalized melodies[4]. Over one hundred electronic instruments are available. The second (see the storyboard in Fig. 3) is a a multimedia Karaoke machine with audiovisual feedback, voice-driven key selection, and performance scoring. In both applications, the user's pitch is displayed in real time, scrolling across the screen as he or she sings into the computer. In the Karaoke demo, the correct pitch is also simultaneously displayed, providing an additional element of embarrassment when the singer misses a note. Both applications run on a laptop with an external microphone.

Interestingly, the real time audiovisual feedback provided by these agents creates a profoundly different user experience than current systems in automatic speech recognition[14]. Unlike dictation programs or dialog managers, our more primitive agents—which only attend to pitch contours—are not designed to replace human operators, but to entertain and amuse in a way that humans cannot. The effect is to enhance the medium of voice, as opposed to highlighting the gap between human and machine performance.

| algorithm | unvoiced in error (%) | voiced in error (%) | gross errors high (%) | low (%) | rms deviation (Hz) |
|---|---|---|---|---|---|
| CPD | 18.11 | 19.89 | 4.09 | 0.64 | 3.60 |
| FBPT | **3.73** | 13.90 | 1.27 | 0.64 | 2.89 |
| HPS | 14.11 | **7.07** | 5.34 | 28.15 | 3.21 |
| IPTA | 9.78 | 17.45 | 1.40 | 0.83 | 3.37 |
| PP | 7.69 | 15.82 | 0.22 | 1.74 | 3.01 |
| SPRD | 4.05 | 15.78 | 0.62 | 2.01 | 2.46 |
| eSPRD | 4.63 | 12.07 | 0.90 | 0.56 | **1.74** |
| ALS | 4.20 | 11.00 | **0.05** | **0.20** | 3.24 |
| CPD | 31.53 | 22.22 | 0.61 | 3.97 | 7.61 |
| FBPT | **3.61** | 12.16 | 0.60 | 3.55 | 7.03 |
| HPS | 19.10 | 21.06 | 0.46 | 1.61 | 5.31 |
| IPTA | 5.70 | 15.93 | 0.53 | 3.12 | 5.35 |
| PP | 6.15 | 13.01 | **0.26** | 3.20 | 6.45 |
| SPRD | 2.35 | 12.16 | 0.39 | 5.56 | 5.51 |
| eSPRD | 2.73 | 9.13 | 0.43 | 0.23 | **5.13** |
| ALS | 4.92 | **5.58** | 0.33 | **0.04** | 6.91 |

Table 1: Evaluations of different pitch tracking algorithms on male speech (top) and female speech (bottom). The algorithms in the table are cepstrum pitch determination (CPD)[9], feature-based pitch tracking (FBPT)[11], harmonic product spectrum (HPS) pitch determination[10, 15], parallel processing (PP) of multiple estimators in the time domain[5], integrated pitch tracking (IPTA)[16], super resolution pitch determination (SRPD)[8], enhanced SRPD (eSRPD)[1], and adaptive least squares (ALS) estimation, as described in this paper. The benchmarks other than ALS were previously reported[1]. The best results in each column are indicated in boldface.

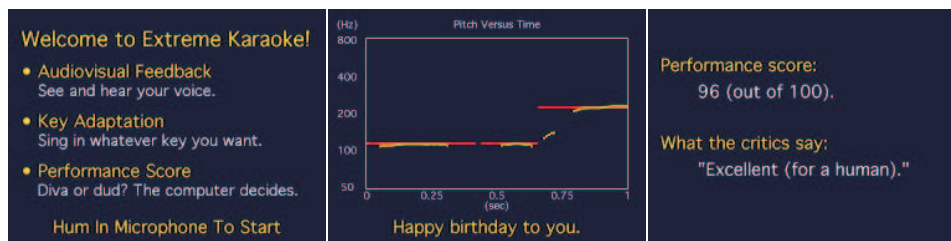

Figure 3: Screen shots from the multimedia Karoake machine with voice-driven key selection, audiovisual feedback, and performance scoring. From left to right: splash screen; singing "happy birthday"; machine evaluation.

## 5 Future work

Voice is the most natural and expressive medium of human communication. Tapping the full potential of this medium remains a grand challenge for researchers in artificial intelligence (AI) and human-computer interaction. In most situations, a speaker's intentions are derived not only from the literal transcription of his speech, but also from prosodic cues, such as pitch, stress, and rhythm. The real time processing of such cues thus represents a fundamental challenge for autonomous, voice-driven agents. Indeed, a machine that could learn from speech as naturally as a newborn infant—responding to prosodic cues but *recognizing in fact no words*—would constitute a genuine triumph of AI.

We are pursuing the ideas in this paper with this vision in mind, looking beyond the immediate applications to voice-to-midi synthesis and audiovisual Karaoke. The algorithm in this paper was purposefully limited to clean speech from non-overlapping speakers. While the algorithm works well in this domain, we view it mainly as a vehicle for experimenting with non-traditional methods that avoid FFTs and autocorrelation and that (ultimately) might be applied to more complicated signals. We have two main goals for future work: first, to add more sophisticated types of human-computer interaction to our voice-driven agents, and second, to incorporate the novel elements of our pitch tracker into a more comprehensive front end for auditory scene analysis[2, 3]. The agents need to be sufficiently complex to engage humans in extended interactions, as well as sufficiently robust to handle corrupted speech and overlapping speakers. From such agents, we expect interesting possibilities to emerge.

## References

[1] P. C. Bagshaw, S. M. Hiller, and M. A. Jack. Enhanced pitch tracking and the processing of f0 contours for computer aided intonation teaching. In *Proceedings of the 3rd European Conference on Speech Communication and Technology*, volume 2, pages 1003–1006, 1993.

[2] A. S. Bregman. *Auditory scene analysis: the perceptual organization of sound*. M.I.T. Press, Cambridge, MA, 1994.

[3] M. Cooke and D. P. W. Ellis. The auditory organization of speech and other sources in listeners and computational models. *Speech Communication*, 35:141–177, 2001.

[4] P. de la Cuadra, A. Master, and C. Sapp. Efficient pitch detection techniques for interactive music. In *Proceedings of the 2001 International Computer Music Conference*, La Habana, Cuba, September 2001.

[5] B. Gold and L. R. Rabiner. Parallel processing techniques for estimating pitch periods of speech in the time domain. *Journal of the Acoustical Society of America*, 46(2,2):442–448, August 1969.

[6] W. M. Hartmann. Pitch, periodicity, and auditory organization. *Journal of the Acoustical Society of America*, 100(6):3491–3502, 1996.

[7] W. Hess. *Pitch Determination of Speech Signals: Algorithms and Devices*. Springer, 1983.

[8] Y. Medan, E. Yair, and D. Chazan. Super resolution pitch determination of speech signals. *IEEE Transactions on Signal Processing*, 39(1):40–48, 1991.

[9] A. M. Noll. Cepstrum pitch determination. *Journal of the Acoustical Society of America*, 41(2):293–309, 1967.

[10] A. M. Noll. Pitch determination of human speech by the harmonic product spectrum, the harmonic sum spectrum, and a maximum likelihood estimate. In *Proceedings of the Symposium on Computer Processing in Communication*, pages 779–798, April 1969.

[11] M. S. Phillips. A feature-based time domain pitch tracker. *Journal of the Acoustical Society of America*, 79:S9–S10, 1985.

[12] J. G. Proakis, C. M. Rader, F. Ling, M. Moonen, I. K. Proudler, and C. L. Nikias. *Algorithms for Statistical Signal Processing*. Prentice Hall, 2002.

[13] L. R. Rabiner. On the use of autocorrelation analysis for pitch determination. *IEEE Transactions on Acoustics, Speech, and Signal Processing*, 25:22–33, 1977.

[14] L. R. Rabiner and B. H. Juang. *Fundamentals of Speech Recognition*. Prentice Hall, Englewoods Cliffs, NJ, 1993.

[15] M. R. Schroeder. Period histogram and product spectrum: new methods for fundamental frequency measurement. *Journal of the Acoustical Society of America*, 43(4):829–834, 1968.

[16] B. G. Secrest and G. R. Doddington. An integrated pitch tracking algorithm for speech systems. In *Proceedings of the 1983 IEEE International Conference on Acoustics, Speech, and Signal Processing*, pages 1352–1355, Boston, 1983.

[17] K. Stevens. *Acoustic Phonetics*. M.I.T. Press, Cambridge, MA, 1999.
